# On the Accuracy of Bounded Rationality: How Far from Optimal Is Fast and Frugal?

**Michael Schmitt**
Ludwig-Marum-Gymnasium
Schlossgartenstraße 11
76327 Pfinztal, Germany
mschmittm@googlemail.com

**Laura Martignon**
Institut für Mathematik und Informatik
Pädagogische Hochschule Ludwigsburg
Reuteallee 46, 71634 Ludwigsburg, Germany
martignon@ph-ludwigsburg.de

## Abstract

Fast and frugal heuristics are well studied models of bounded rationality. Psychological research has proposed the take-the-best heuristic as a successful strategy in decision making with limited resources. Take-the-best searches for a sufficiently good ordering of cues (features) in a task where objects are to be compared lexicographically. We investigate the complexity of the problem of approximating optimal cue permutations for lexicographic strategies. We show that no efficient algorithm can approximate the optimum to within any constant factor, if P ≠ NP. We further consider a greedy approach for building lexicographic strategies and derive tight bounds for the performance ratio of a new and simple algorithm. This algorithm is proven to perform better than take-the-best.

## 1 Introduction

In many circumstances the human mind has to make decisions when time and knowledge are limited. Cognitive psychology categorizes human judgments made under such constraints as being boundedly rational if they are "satisficing" (Simon, 1982) or, more generally, if they do not fall too far behind the rational standards. A class of models for human reasoning studied in the context of bounded rationality consists of simple algorithms termed "fast and frugal heuristics". These were the topic of major psychological research (Gigerenzer and Goldstein, 1996; Gigerenzer et al., 1999). Great efforts have been put into testing these heuristics by empirical means in experiments with human subjects (Bröder, 2000; Bröder and Schiffer, 2003; Lee and Cummins, 2004; Newell and Shanks, 2003; Newell et al., 2003; Slegers et al., 2000) or in simulations on computers (Bröder, 2002; Hogarth and Karelaia, 2003; Nellen, 2003; Todd and Dieckmann, 2005). (See also the discussion and controversies documented in the open peer commentaries on Todd and Gigerenzer, 2000.)

Among the fast and frugal heuristics there is an algorithm called "take-the-best" (TTB) that is considered a process model for human judgments based on one-reason decision making. Which of the two cities has a larger population: (a) Düsseldorf (b) Hamburg? This is the task originally studied by Gigerenzer and Goldstein (1996) where German cities with a population of more than 100,000 inhabitants had to be compared. The available information on each city consists of the values of nine binary cues, or attributes, indicating

|  | Soccer Team | State Capital | License Plate |
|---|---|---|---|
| Hamburg | 1 | 1 | 0 |
| Essen | 0 | 0 | 1 |
| Düsseldorf | 0 | 1 | 1 |
| Validity | 1 | 1/2 | 0 |

Table 1: Part of the German cities task of Gigerenzer and Goldstein (1996). Shown are profiles and validities of three cues for three cities. Cue validities are computed from the data as given here. The original data has different validities but the same cue ranking.

presence or absence of a feature. The cues being used are, for instance, whether the city is a state capital, whether it is indicated on car license plates by a single letter, or whether it has a soccer team in the national league. The judgment which city is larger is made on the basis of the two binary vectors, or cue profiles, representing the two cities. TTB performs a lexicographic strategy, comparing the cues one after the other and using the first cue that discriminates as the one reason to yield the final decision. For instance, if one city has a university and the other does not, TTB would infer that the first city is larger than the second. If the cue values of both cities are equal, the algorithm passes on to the next cue.

TTB examines the cues in a certain order. Gigerenzer and Goldstein (1996) introduced ecological validity as a numerical measure for ranking the cues. The validity of a cue is a real number in the interval $[0,1]$ that is computed in terms of the known outcomes of paired comparisons. It is defined as the number of pairs the cue discriminates correctly (i.e., where it makes a correct inference) divided by the number of pairs it discriminates (i.e., where it makes an inference, be it right or wrong). TTB always chooses a cue with the highest validity, that is, it "takes the best" among those cues not yet considered. Table 1 shows cue profiles and validities for three cities. The ordering defined by the size of their population is given by

$$\{\langle\, \text{Düsseldorf}, \text{Essen}\,\rangle, \langle\, \text{Düsseldorf}, \text{Hamburg}\,\rangle, \langle\, \text{Essen}, \text{Hamburg}\,\rangle\},$$

where a pair $\langle a, b \rangle$ indicates that $a$ has less inhabitants than $b$. As an example for calculating the validity, the state-capital cue distinguishes the first and the third pair but is correct only on the latter. Hence, its validity has value $1/2$.

The order in which the cues are ranked is crucial for success or failure of TTB. In the example of Düsseldorf and Hamburg, the car-license-plate cue would yield that Düsseldorf (D) is larger than Hamburg (HH), whereas the soccer-team cue would correctly favor Hamburg. Thus, how successful a lexicographic strategy is in a comparison task consisting of a partial ordering of cue profiles depends on how well the cue ranking minimizes the number of incorrect comparisons. Specifically, the accuracy of TTB relies on the degree of optimality achieved by the ranking according to decreasing cue validities. For TTB and the German cities task, computer simulations have shown that TTB discriminates at least as accurate as other models (Gigerenzer and Goldstein, 1996; Gigerenzer et al., 1999; Todd and Dieckmann, 2005). TTB made as many correct inferences as standard algorithms proposed by cognitive psychology and even outperformed some of them.

Partial results concerning the accuracy of TTB compared to the accuracy of other strategies have been obtained analytically by Martignon and Hoffrage (2002). Here we subject the problem of finding optimal cue orderings to a rigorous theoretical analysis employing methods from the theory of computational complexity (Ausiello et al., 1999). Obviously, TTB runs in polynomial time. Given a list of ordered pairs, it computes all cue validities in polynomially many computing steps in terms of the size of the list. We define the optimization problem MINIMUM INCORRECT LEXICOGRAPHIC STRATEGY as the task of minimizing the number of incorrect inferences for the lexicographic strategy on a given list of pairs. We show that, unless P = NP, there is no polynomial-time approximation algo-

rithm that computes solutions for MINIMUM INCORRECT LEXICOGRAPHIC STRATEGY that are only a constant factor worse than the optimum, unless $P = NP$. This means that the approximating factor, or performance ratio, must grow with the size of the problem.

As an extension of TTB we consider an algorithm for finding cue orderings that was called "TTB by Conditional Validity" in the context of bounded rationality. It is based on the greedy method, a principle widely used in algorithm design. This greedy algorithm runs in polynomial time and we derive tight bounds for it, showing that it approximates the optimum with a performance ratio proportional to the number of cues. An important consequence of this result is a guarantee that for those instances that have a solution that discriminates all pairs correctly, the greedy algorithm always finds a permutation attaining this minimum. We are not aware that this quality has been established for any of the previously studied heuristics for paired comparison. In addition, we show that TTB does not have this property, concluding that the greedy method of constructing cue permutations performs provably better than TTB. For a more detailed account and further results we refer to the complete version of this work (Schmitt and Martignon, 2006).

## 2 Lexicographic Strategies

A *lexicographic strategy* is a method for comparing elements of a set $B \subseteq \{0,1\}^n$. Each component $1, \ldots, n$ of these vectors is referred to as a *cue*. Given $a, b \in B$, where $a = (a_1, \ldots, a_n)$ and $b = (b_1, \ldots, b_n)$, the lexicographic strategy searches for the smallest cue index $i \in \{1, \ldots, n\}$ such that $a_i$ and $b_i$ are different. The strategy then outputs one of " $<$ " or " $>$ " according to whether $a_i < b_i$ or $a_i > b_i$ assuming the usual order $0 < 1$ of the truth values. If no such cue exists, the strategy returns " $=$ ". Formally, let diff $: B \times B \to \{1, \ldots, n+1\}$ be the function where $\mathrm{diff}(a,b)$ is the smallest cue index on which $a$ and $b$ are different, or $n+1$ if they are equal, that is,

$$\mathrm{diff}(a,b) \quad = \quad \min\{\{i : a_i \neq b_i\} \cup \{n+1\}\}.$$

Then, the function $S : B \times B \to \{\text{"} < \text{"}, \text{"} = \text{"}, \text{"} > \text{"}\}$ computed by the lexicographic strategy is

$$S(a,b) \quad = \quad \begin{cases} \text{"} < \text{"} & \text{if } \mathrm{diff}(a,b) \leq n \text{ and } a_{\mathrm{diff}(a,b)} < b_{\mathrm{diff}(a,b)}, \\ \text{"} > \text{"} & \text{if } \mathrm{diff}(a,b) \leq n \text{ and } a_{\mathrm{diff}(a,b)} > b_{\mathrm{diff}(a,b)}, \\ \text{"} = \text{"} & \text{otherwise.} \end{cases}$$

Lexicographic strategies may take into account that the cues come in an order that is different from $1, \ldots, n$. Let $\pi : \{1, \ldots, n\} \to \{1, \ldots, n\}$ be a permutation of the cues. It gives rise to a mapping $\overline{\pi} : \{0,1\}^n \to \{0,1\}^n$ that permutes the components of Boolean vectors by $\overline{\pi}(a_1, \ldots, a_n) = (a_{\pi(1)}, \ldots, a_{\pi(n)})$. As $\overline{\pi}$ is uniquely defined given $\pi$, we simplify the notation and write also $\pi$ for $\overline{\pi}$. The *lexicographic strategy under cue permutation* $\pi$ passes through the cues in the order $\pi(1), \ldots, \pi(n)$, that is, it computes the function $S_\pi : B \times B \to \{\text{"} < \text{"}, \text{"} = \text{"}, \text{"} > \text{"}\}$ defined as

$$S_\pi(a,b) \quad = \quad S(\pi(a), \pi(b)).$$

The problem we study is that of finding a cue permutation that minimizes the number of incorrect comparisons in a given list of element pairs using the lexicographic strategy. An instance of this problem consists of a set $B$ of elements and a set of pairs $L \subseteq B \times B$. Each pair $\langle a, b \rangle \in L$ represents an inequality $a \leq b$. Given a cue permutation $\pi$, we say that the lexicographic strategy under $\pi$ *infers* the pair $\langle a, b \rangle$ *correctly* if $S_\pi(a,b) \in \{\text{"} < \text{"}, \text{"} = \text{"}\}$, otherwise the inference is incorrect. The task is to find a permutation $\pi$ such that the number of incorrect inferences in $L$ using $S_\pi$ is minimal, that is, a permutation $\pi$ that minimizes

$$\mathrm{INCORRECT}(\pi, L) \quad = \quad |\{\langle a, b \rangle \in L : S_\pi(a,b) = \text{"} > \text{"}\}|.$$

## 3 Approximability of Optimal Cue Permutations

A large class of optimization problems, denoted APX, can be solved efficiently if the solution is required to be only a constant factor worse than the optimum (see, e.g., Ausiello et al., 1999). Here, we prove that, if $P \neq NP$, there is no polynomial-time algorithm whose solutions yield a number of incorrect comparisons that is by at most a constant factor larger than the minimal number possible. It follows that the problem of approximating the optimal cue permutation is even harder than any problem in APX. The optimization problem is formally stated as follows.

> MINIMUM INCORRECT LEXICOGRAPHIC STRATEGY
> Instance: A set $B \subseteq \{0,1\}^n$ and a set $L \subseteq B \times B$.
> Solution: A permutation $\pi$ of the cues of $B$.
> Measure: The number of incorrect inferences in $L$ for the lexicographic strategy under cue permutation $\pi$, that is, INCORRECT$(\pi, L)$.

Given a real number $r > 0$, an algorithm is said to approximate MINIMUM INCORRECT LEXICOGRAPHIC STRATEGY to within a factor of $r$ if for every instance $(B, L)$ the algorithm returns a permutation $\pi$ such that

$$\text{INCORRECT}(\pi, L) \quad \leq \quad r \cdot \text{opt}(L),$$

where opt$(L)$ is the minimal number of incorrect comparisons achievable on $L$ by any permutation. The factor $r$ is also known as the performance ratio of the algorithm. The following optimization problem plays a crucial role in the derivation of the lower bound for the approximability of MINIMUM INCORRECT LEXICOGRAPHIC STRATEGY.

> MINIMUM HITTING SET
> Instance: A collection $C$ of subsets of a finite set $U$.
> Solution: A hitting set for $C$, that is, a subset $U' \subseteq U$ such that $U'$ contains at least one element from each subset in $C$.
> Measure: The cardinality of the hitting set, that is, $|U'|$.

MINIMUM HITTING SET is equivalent to MINIMUM SET COVER. Bellare et al. (1993) have shown that MINIMUM SET COVER cannot be approximated in polynomial time to within any constant factor, unless $P = NP$. Thus, if $P \neq NP$, MINIMUM HITTING SET cannot be approximated in polynomial time to within any constant factor as well.

**Theorem 1.** *For every $r$, there is no polynomial-time algorithm that approximates* MINIMUM INCORRECT LEXICOGRAPHIC STRATEGY *to within a factor of $r$, unless* $P = NP$.

*Proof.* We show that the existence of a polynomial-time algorithm that approximates MINIMUM INCORRECT LEXICOGRAPHIC STRATEGY to within some constant factor implies the existence of a polynomial-time algorithm that approximates MINIMUM HITTING SET to within the same factor. Then the statement follows from the equivalence of MINIMUM HITTING SET with MINIMUM SET COVER and the nonapproximability of the latter (Bellare et al., 1993). The main part of the proof consists in establishing a specific approximation preserving reduction, or AP-reduction, from MINIMUM HITTING SET to MINIMUM INCORRECT LEXICOGRAPHIC STRATEGY. (See Ausiello et al., 1999, for a definition of the AP-reduction.).

We first define a function $f$ that is computable in polynomial time and maps each instance of MINIMUM HITTING SET to an instance of MINIMUM INCORRECT LEXICOGRAPHIC STRATEGY. Let $\mathbf{1}$ denote the $n$-bit vector with a 1 everywhere and $\mathbf{1}_{i_1,\ldots,i_\ell}$ the vector with 0 in positions $i_1, \ldots, i_\ell$ and 1 elsewhere. Given the collection $C$ of subsets of the set $U = \{u_1, \ldots, u_n\}$, the function $f$ maps $C$ to $(B, L)$, where $B \subseteq \{0,1\}^{n+1}$ is defined as follows:

1. Let $(\mathbf{1}, 0) \in B$.
2. For $i = 1, \ldots, n$, let $(\mathbf{1}_i, 1) \in B$.
3. For every $\{u_{i_1}, \ldots, u_{i_\ell}\} \in C$, let $(\mathbf{1}_{i_1, \ldots, i_\ell}, 1) \in B$.

Further, the set $L$ is constructed as

$$L = \{\langle (\mathbf{1}, 0), (\mathbf{1}_i, 1) \rangle : i = 1, \ldots, n\} \cup \{\langle (\mathbf{1}_{i_1, \ldots, i_\ell}, 1), (\mathbf{1}, 0) \rangle : \{u_{i_1}, \ldots, u_{i_\ell}\} \in C\}. \quad (1)$$

In the following, a pair from the first and second set on the right-hand side of equation (1) is referred to as an element pair and a subset pair, respectively. Obviously, the function $f$ is computable in polynomial time. It has the following property.

**Claim 1.** *Let $f(C) = (B, L)$. If $C$ has a hitting set of cardinality $k$ or less then $f(C)$ has a cue permutation $\pi$ where* INCORRECT$(\pi, L) \leq k$.

To prove this, assume without loss of generality that $C$ has a hitting set $U'$ of cardinality exactly $k$, say $U' = \{u_{j_1}, \ldots, u_{j_k}\}$, and let $U \setminus U' = \{u_{j_{k+1}}, \ldots, u_{j_n}\}$. Then the cue permutation

$$j_1, \ldots, j_k, n + 1, j_{k+1}, \ldots, j_n.$$

results in no more than $k$ incorrect inferences in $L$. Indeed, consider an arbitrary subset pair $\langle (\mathbf{1}_{i_1, \ldots, i_\ell}, 1), (\mathbf{1}, 0) \rangle$. To not be an error, one of $i_1, \ldots, i_\ell$ must occur in the hitting set $j_1, \ldots, j_k$. Hence, the first cue that distinguishes this pair has value 0 in $(\mathbf{1}_{i_1, \ldots, i_\ell}, 1)$ and value 1 in $(\mathbf{1}, 0)$, resulting in a correct comparison. Further, let $\langle (\mathbf{1}, 0), (\mathbf{1}_i, 1) \rangle$ be an element pair with $u_i \notin U'$. This pair is distinguished correctly by cue $n + 1$. Finally, each element pair $\langle (\mathbf{1}, 0), (\mathbf{1}_i, 1) \rangle$ with $u_i \in U'$ is distinguished by cue $i$ with a result that disagrees with the ordering given by $L$. Thus, only element pairs with $u_i \in U'$ yield incorrect comparisons and no subset pair. Hence, the number of incorrect inferences is not larger than $|U'|$.

Next, we define a polynomial-time computable function $g$ that maps each collection $C$ of subsets of a finite set $U$ and each cue permutation $\pi$ for $f(C)$ to a subset of $U$. Given that $f(C) = (B, L)$, the set $g(C, \pi) \subseteq U$ is defined as follows:

1. For every element pair $\langle (\mathbf{1}, 0), (\mathbf{1}_i, 1) \rangle \in L$ that is compared incorrectly by $\pi$, let $u_i \in g(C, \pi)$.

2. For every subset pair $\langle (\mathbf{1}_{i_1, \ldots, i_\ell}, 1), (\mathbf{1}, 0) \rangle \in L$ that is compared incorrectly by $\pi$, let one of the elements $u_{i_1}, \ldots, u_{i_\ell} \in g(C, \pi)$.

Clearly, the function $g$ is computable in polynomial time. It satisfies the following condition.

**Claim 2.** *Let $f(C) = (B, L)$. If* INCORRECT$(\pi, L) \leq k$ *then $g(C, \pi)$ is a hitting set of cardinality $k$ or less for $C$.*

Obviously, if INCORRECT$(\pi, L) \leq k$ then $g(C, \pi)$ has cardinality at most $k$. To show that it is a hitting set, assume the subset $\{u_{i_1}, \ldots, u_{i_\ell}\} \in C$ is not hit by $g(C, \pi)$. Then neither of $u_{i_1}, \ldots, u_{i_\ell}$ is in $g(C, \pi)$. Hence, we have correct comparisons for the element pairs corresponding to $u_{i_1}, \ldots, u_{i_\ell}$ and for the subset pair corresponding to $\{u_{i_1}, \ldots, u_{i_\ell}\}$. As the subset pair is distinguished correctly, one of the cues $i_1, \ldots, i_\ell$ must be ranked before cue $n + 1$. But then at least one of the element pairs for $u_{i_1}, \ldots, u_{i_\ell}$ yields an incorrect comparison. This contradicts the assertion that the comparisons for these element pairs are all correct. Thus, $g(C, \pi)$ is a hitting set and the claim is established.

Assume now that there exists a polynomial-time algorithm $A$ that approximates MINIMUM INCORRECT LEXICOGRAPHIC STRATEGY to within a factor of $r$. Consider the algorithm that, for a given instance $C$ of MINIMUM HITTING SET as input, calls algorithm $A$ with input $(B, L) = f(C)$, and returns $g(C, \pi)$ where $\pi$ is the output provided by $A$. Clearly, this new algorithm runs in polynomial time. We show that it approximates MINIMUM

---
**Algorithm 1** GREEDY CUE PERMUTATION
---
**Input:** a set $B \subseteq \{0,1\}^n$ and a set $L \subseteq B \times B$
**Output:** a cue permutation $\pi$ for $n$ cues
  $I := \{1, \ldots, n\}$;
  **for** $i = 1, \ldots, n$ **do**
    let $j \in I$ be a cue where $\text{INCORRECT}(j, L) = \min_{j' \in I} \text{INCORRECT}(j', L)$;
    $\pi(i) := j$;
    $I := I \setminus \{j\}$;
    $L := L \setminus \{\langle a, b \rangle : a_j \neq b_j\}$
  **end for**.
---

HITTING SET to within a factor of $r$. By the assumed approximation property of algorithm $A$, we have

$$\text{INCORRECT}(\pi, L) \quad \leq \quad r \cdot \text{opt}(L).$$

Together with Claim 2, this implies that $g(\pi, C)$ is a hitting set for $C$ satisfying

$$|g(C, \pi)| \quad \leq \quad r \cdot \text{opt}(L).$$

From Claim 1 we obtain $\text{opt}(L) \leq \text{opt}(C)$ and, thus,

$$|g(C, \pi)| \quad \leq \quad r \cdot \text{opt}(C).$$

Thus, the proposed algorithm for MINIMUM HITTING SET violates the approximation lower bound that holds for this problem under the assumption $P \neq NP$. This proves the statement of the theorem. $\square$

## 4 Greedy Approximation of Optimal Cue Permutations

The so-called greedy approach to the solution of an approximation problem is helpful when it is not known which algorithm performs best. It is a simple heuristic that in practice often provides satisfactory solutions in many situations. The algorithm GREEDY CUE PERMU-TATION that we introduce here is based on the greedy method. The idea is to select the first cue according to which single cue makes a minimum number of incorrect inferences (choosing one arbitrarily if there are two or more). After that the algorithm removes those pairs that are distinguished by the selected cue, which is reasonable as the distinctions drawn by this cue cannot be undone by later cues. This procedure is then repeated on the set of pairs left. The description of GREEDY CUE PERMUTATION is given as Algorithm 1. It employs an extension of the function INCORRECT applicable to single cues, such that for a cue $i$ we have

$$\text{INCORRECT}(i, L) \quad = \quad |\{\langle a, b \rangle \in L : a_i > b_i\}|.$$

It is evident that Algorithm 1 runs in polynomial time, but how good is it? The least one should demand from a good heuristic is that, whenever a minimum of zero is attainable, it finds such a solution. This is indeed the case with GREEDY CUE PERMUTATION as we show in the following result. Moreover, it asserts a general performance ratio for the approximation of the optimum.

**Theorem 2.** *The algorithm* GREEDY CUE PERMUTATION *approximates* MINIMUM IN-CORRECT LEXICOGRAPHIC STRATEGY *to within a factor of* $n$*, where* $n$ *is the number of cues. In particular, it always finds a cue permutation with no incorrect inferences if one exists.*

*Proof.* We show by induction on $n$ that the permutation returned by the algorithm makes a number of incorrect inferences no larger than $n \cdot \text{opt}(L)$. If $n = 1$, the optimal cue

$$\langle\ 001\ ,\ 010\ \rangle$$
$$\langle\ 010\ ,\ 100\ \rangle$$
$$\langle\ 010\ ,\ 101\ \rangle$$
$$\langle\ 100\ ,\ 111\ \rangle$$

Figure 1: A set of lexicographically ordered pairs with nondecreasing cue validities $(1, 1/2,$ and $2/3)$. The cue ordering of TTB $(1, 3, 2)$ causes an incorrect inference on the first pair. By Theorem 2, GREEDY CUE PERMUTATION finds the lexicographic ordering.

permutation is definitely found. Let $n > 1$. Clearly, as the incorrect inferences of a cue cannot be reversed by other cues, there is a cue $j$ with

$$\text{INCORRECT}(j, L) \leq \text{opt}(L).$$

The algorithm selects such a cue in the first round of the loop. During the rest of the rounds, a permutation of $n-1$ cues is constructed for the set of remaining pairs. Let $j$ be the cue that is chosen in the first round, $I' = \{1, \ldots, j-1, j+1, \ldots, n\}$, and $L' = L \setminus \{\langle a, b \rangle : a_j \neq b_j\}$. Further, let $\text{opt}_{I'}(L')$ denote the minimum number of incorrect inferences taken over the permutations of $I'$ on the set $L'$. Then, we observe that

$$\text{opt}(L) \geq \text{opt}(L') = \text{opt}_{I'}(L').$$

The inequality is valid because of $L \supseteq L'$. (Note that $\text{opt}(L')$ refers to the minimum taken over the permutations of all cues.) The equality holds as cue $j$ does not distinguish any pair in $L'$. By the induction hypothesis, rounds 2 to $n$ of the loop determine a cue permutation $\pi'$ with $\text{INCORRECT}(\pi', L') \leq (n-1) \cdot \text{opt}_{I'}(L')$. Thus, the number of incorrect inferences made by the permutation $\pi$ finally returned by the algorithm satisfies

$$\text{INCORRECT}(\pi, L) \quad \leq \quad \text{INCORRECT}(j, L) + (n-1) \cdot \text{opt}_{I'}(L'),$$

which is, by the inequalities derived above, not larger than $\text{opt}(L) + (n-1) \cdot \text{opt}(L)$ as stated. □

**Corollary 3.** *On inputs that have a cue ordering without incorrect comparisons under the lexicographic strategy,* GREEDY CUE PERMUTATION *can be better than TTB.*

*Proof.* Figure 1 shows a set of four lexicographically ordered pairs. According to Theorem 2, GREEDY CUE PERMUTATION comes up with the given permutation of the cues. The validities are $1, 1/2$, and $2/3$. Thus, TTB ranks the cues as $1, 3, 2$ whereupon the first pair is inferred incorrectly. □

Finally, we consider lower bounds on the performance ratio of GREEDY CUE PERMUTATION. The proof of this claim is omitted here.

**Theorem 4.** *The performance ratio of* GREEDY CUE PERMUTATION *is at least* $\max\{n/2, |L|/2\}$.

## 5  Conclusions

The result that the optimization problem MINIMUM INCORRECT LEXICOGRAPHIC STRATEGY cannot be approximated in polynomial time to within any constant factor answers a long-standing question of psychological research into models of bounded rationality: How accurate are fast and frugal heuristics? It follows that no fast, that is, polynomial-time, algorithm can approximate the optimum well, under the widely accepted assumption that $P \neq NP$. A further question is concerned with a specific fast and frugal heuristic: How accurate is TTB? The new algorithm GREEDY CUE PERMUTATION has been shown to perform provably better than TTB. In detail, it always finds accurate solutions when they exist, in contrast to TTB. With this contribution we pose a challenge to cognitive psychology: to study the relevance of the greedy method as a model for bounded rationality.

**Acknowledgment.** The first author has been supported in part by the Deutsche Forschungsgemeinschaft (DFG).

## References

Ausiello, G., Crescenzi, P., Gambosi, G., Kann, V., Marchetti-Spaccamela, A., and Protasi, M. (1999). *Complexity and Approximation: Combinatorial Problems and Their Approximability Properties*. Springer-Verlag, Berlin.

Bellare, M., Goldwasser, S., Lund, C., and Russell, A. (1993). Efficient probabilistically checkable proofs and applications to approximation. In *Proceedings of the 25th Annual ACM Symposium on Theory of Computing*, pages 294–304. ACM Press, New York, NY.

Bröder, A. (2000). Assessing the empirical validity of the "take-the-best" heuristic as a model of human probabilistic inference. *Journal of Experimental Psychology: Learning, Memory, and Cognition*, 26:1332–1346.

Bröder, A. (2002). Take the best, Dawes' rule, and compensatory decision strategies: A regression-based classification method. *Quality & Quantity*, 36:219–238.

Bröder, A. and Schiffer, S. (2003). Take the best versus simultaneous feature matching: Probabilistic inferences from memory and effects of representation format. *Journal of Experimental Psychology: General*, 132:277–293.

Gigerenzer, G. and Goldstein, D. G. (1996). Reasoning the fast and frugal way: Models of bounded rationality. *Psychological Review*, 103:650–669.

Gigerenzer, G., Todd, P. M., and the ABC Research Group (1999). *Simple Heuristics That Make Us Smart*. Oxford University Press, New York, NY.

Hogarth, R. M. and Karelaia, N. (2003). "Take-the-best" and other simple strategies: Why and when they work "well" in binary choice. DEE Working Paper 709, Universitat Pompeu Fabra, Barcelona.

Lee, M. D. and Cummins, T. D. R. (2004). Evidence accumulation in decision making: Unifying the "take the best" and the "rational" models. *Psychonomic Bulletin & Review*, 11:343–352.

Martignon, L. and Hoffrage, U. (2002). Fast, frugal, and fit: Simple heuristics for paired comparison. *Theory and Decision*, 52:29–71.

Nellen, S. (2003). The use of the "take the best" heuristic under different conditions, modeled with ACT-R. In Detje, F., Dörner, D., and Schaub, H., editors, *Proceedings of the Fifth International Conference on Cognitive Modeling*, pages 171–176, Universitätsverlag Bamberg, Bamberg.

Newell, B. R. and Shanks, D. R. (2003). Take the best or look at the rest? Factors influencing "One-Reason" decision making. *Journal of Experimental Psychology: Learning, Memory, and Cognition*, 29:53–65.

Newell, B. R., Weston, N. J., and Shanks, D. R. (2003). Empirical tests of a fast-and-frugal heuristic: Not everyone "takes-the-best". *Organizational Behavior and Human Decision Processes*, 91:82–96.

Schmitt, M. and Martignon, L. (2006). On the complexity of learning lexicographic strategies. *Journal of Machine Learning Research*, 7(Jan):55–83.

Simon, H. A. (1982). *Models of Bounded Rationality, Volume 2*. MIT Press, Cambridge, MA.

Slegers, D. W., Brake, G. L., and Doherty, M. E. (2000). Probabilistic mental models with continuous predictors. *Organizational Behavior and Human Decision Processes*, 81:98–114.

Todd, P. M. and Dieckmann, A. (2005). Heuristics for ordering cue search in decision making. In Saul, L. K., Weiss, Y., and Bottou, L., editors, *Advances in Neural Information Processing Systems 17*, pages 1393–1400. MIT Press, Cambridge, MA.

Todd, P. M. and Gigerenzer, G. (2000). Précis of "Simple Heuristics That Make Us Smart". *Behavioral and Brain Sciences*, 23:727–741.